# Using Pairs of Data-Points to Define Splits for Decision Trees

**Geoffrey E. Hinton**
Department of Computer Science
University of Toronto
Toronto, Ontario, M5S 1A4, Canada
hinton@cs.toronto.edu

**Michael Revow**
Department of Computer Science
University of Toronto
Toronto, Ontario, M5S 1A4, Canada
revow@cs.toronto.edu

## Abstract

Conventional binary classification trees such as CART either split the data using axis-aligned hyperplanes or they perform a computationally expensive search in the continuous space of hyperplanes with unrestricted orientations. We show that the limitations of the former can be overcome without resorting to the latter. For every pair of training data-points, there is one hyperplane that is orthogonal to the line joining the data-points and bisects this line. Such hyperplanes are plausible candidates for splits. In a comparison on a suite of 12 datasets we found that this method of generating candidate splits outperformed the standard methods, particularly when the training sets were small.

## 1 Introduction

Binary decision trees come in many flavours, but they all rely on splitting the set of k-dimensional data-points at each internal node into two disjoint sets. Each split is usually performed by projecting the data onto some direction in the k-dimensional space and then thresholding the scalar value of the projection. There are two commonly used methods of picking a projection direction. The simplest method is to restrict the allowable directions to the $k$ axes defined by the data. This is the default method used in CART [1]. If this set of directions is too restrictive, the usual alternative is to search general directions in the full $k$-dimensional space or general directions in a space defined by a subset of the $k$ axes.

Projections onto one of the $k$ axes defined by the the data have many advantages

over projections onto a more general direction:

1. It is very efficient to perform the projection for each of the data-points. We simply ignore the values of the data-point on the other axes.

2. For $N$ data-points, it is feasible to consider all possible axis-aligned projections and thresholds because there are only $k$ possible projections and for each of these there are at most $N-1$ threshold values that yield different splits. Selecting from a fixed set of projections and thresholds is simpler than searching the $k$-dimensional continuous space of hyperplanes that correspond to unrestricted projections and thresholds.

3. Since a split is selected from only about $Nk$ candidates, it takes only about $\log_2 N + \log_2 k$ bits to define the split. So it should be possible to use many more of these axis-aligned splits before overfitting occurs than if we use more general hyperplanes. If the data-points are in general position, each subset of size $k$ defines a different hyperplane so there are $N!/k!(N-k)!$ distinctly different hyperplanes and if $k << N$ it takes approximately $k\log_2 N$ bits to specify one of them.

For some datasets, the restriction to axis-aligned projections is too limiting. This is especially true for high-dimensional data, like images, in which there are strong correlations between the intensities of neighbouring pixels. In such cases, many axis-aligned boundaries may be required to approximate a planar boundary that is not axis-aligned, so it is natural to consider unrestricted projections and some versions of the CART program allow this. Unfortunately this greatly increases the computational burden and the search may get trapped in local minima. Also significant care must be exercised to avoid overfitting. There is, however, an intermediate approach which allows the projections to be non-axis-aligned but preserves all three of the attractive properties of axis-aligned projections: It is trivial to decide which side of the resulting hyperplane a given data-point lies on; the hyperplanes can be selected from a modest-sized set of sensible candidates; and hence many splits can be used before overfitting occurs because only a few bits are required to specify each split.

## 2  Using two data-points to define a projection

Each pair of data-points defines a direction in the data space. This direction is a plausible candidate for a projection to be used in splitting the data, especially if it is a classification task and the two data-points are in different classes. For each such direction, we could consider all of the $N-1$ possible thresholds that would give different splits, or, to save time and reduce complexity, we could only consider the threshold value that is halfway between the two data-points that define the projection. If we use this threshold value, each pair of data-points defines exactly one hyperplane and we call the two data-points the "poles" of this hyperplane.

For a general $k$-dimensional hyperplane it requires $O(k)$ operations to decide whether a data-point, $C$, is on one side or the other. But we can save a factor of $k$ by using hyperplanes defined by pairs of data-points. If we already know the distances of $C$ from each of the two poles, $A, B$ then we only need to compare

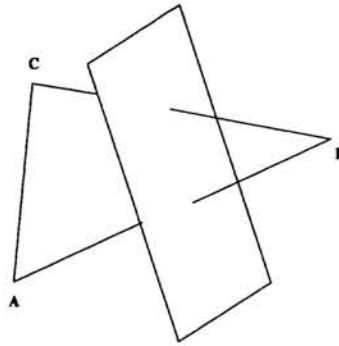

Figure 1: A hyperplane orthogonal to the line joining points $A$ and $B$. We can quickly determine on which side a test point, $C$, lies by comparing the distances $AC$ and $BC$.

these two distances (see figure 1).[1] So if we are willing to do $O(kN^2)$ operations to compute all the pairwise distances between the data-points, we can then decide in constant time which side of the hyperplane a point lies on.

As we are building the decision tree, we need to compute the gain in performance from using each possible split at each existing terminal node. Since all the terminal nodes combined contain $N$ data-points and there are $N(N-1)/2$ possible splits[2] this takes time $O(N^3)$ instead of $O(kN^3)$. So the work in computing all the pairwise distances is trivial compared with the savings.

Using the Minimum Description Length framework, it is clear that pole-pair splits can be described very cheaply, so a lot of them can be used before overfitting occurs. When applying MDL to a supervised learning task we can assume that the receiver gets to see the input vectors for free. It is only the output vectors that need to be communicated. So if splits are selected from a set of $N(N-1)/2$ possibilities that is determined by the input vectors, it takes only about $2\log_2 N$ bits to communicate a split to a receiver. Even if we allow all $N-1$ possible threshold values along the projection defined by two data-points, it takes only about $3\log_2 N$ bits. So the number of these splits that can be used before overfitting occurs should be greater by a factor of about $k/2$ or $k/3$ than for general hyperplanes. Assuming that $k << N$, the same line of argument suggests that even more axis-aligned planes can be used, but only by a factor of about 2 or 3.

To summarize, the hyperplanes planes defined by pairs of data-points are computationally convenient and seem like natural candidates for good splits. They overcome the major weakness of axis-aligned splits and, because they can be specified in a modest number of bits, they may be more effective than fully general hyperplanes when the training set is small.

## 3   Building the decision tree

We want to compare the "pole-pair" method of generating candidate hyperplanes with the standard axis-aligned method and the method that uses unrestricted hyperplanes. We can see no reason to expect strong interactions between the method of building the tree and the method of generating the candidate hyperplanes, but to minimize confounding effects we always use exactly the same method of building the decision tree.

We faithfully followed the method described in [1], except for a small modification where the code that was kindly supplied by Leo Breiman used a slightly different method for determining the amount of pruning.

Training a decision tree involves two distinct stages. In the first stage, nodes are repeatedly split until each terminal node is "pure" which means that all of its datapoints belong to the same class. The pure tree therefore fits the training data perfectly. A node is split by considering all candidate decision planes and choosing the one that maximizes the decrease in impurity. Breiman *et. al* recommend using the *Gini* index to measure impurity.[3] If $p(j|t)$ is the probability of class $j$ at node $t$, then the Gini index is $1 - \sum_j p^2(j|t)$.

Clearly the tree obtained at the end of the first stage will overfit the data and so in the second stage the tree is pruned by recombining nodes. For a tree, $T_i$, with $|T_i|$ terminal nodes we consider the regularized cost:

$$C = E + \alpha |T_i| \tag{1}$$

where $E$ is the classification error and $\alpha$ is a pruning parameter. In "weakest-link" pruning the terminal nodes are eliminated in the order which keeps (1) minimal as $\alpha$ increases. This leads to a particular sequence, $T = \{T_1, T_2, ...T_k\}$ of subtrees, in which $|T_1| > |T_2|... > |T_k|$. We call this the "main" sequence of subtrees because they are trained on *all* of the training data.

The last remaining issue to be resolved is which tree in the main sequence to use. The simplest method is to use a separate validation set and choose the tree size that gives best classification on it. Unfortunately, many of the datasets we used were too small to hold back a reserved validation set. So we always used 10-fold cross validation to pick the size of the tree. We first grew 10 different subsidiary trees until their terminal nodes were pure, using 9/10 of the data for training each of them. Then we pruned back each of these pure subsidiary trees, as above, producing 10 sequences of subsidiary subtrees. These subsidiary sequences could then be used for estimating the performance of each subtree in the main sequence. For each of the main subtrees, $T_i$, we found the largest tree in each subsidiary sequence that was no larger than $T_i$ and estimated the performance of $T_i$ to be the average of the performance achieved by each subsidiary subtree on the 1/10 of the data that was not used for training that subsidiary tree. We then chose the $T_i$ that achieved the best performance estimate and used it on the test set[4]. Results are expressed as

|                | IR  | TR  | LV  | DB  | BC  | GL  | VW  | WN  | VH  | WV   | IS  | SN  |
|----------------|-----|-----|-----|-----|-----|-----|-----|-----|-----|------|-----|-----|
| Size (N)       | 150 | 215 | 345 | 768 | 683 | 163 | 990 | 178 | 846 | 2100 | 351 | 208 |
| Classes (c)    | 3   | 3   | 2   | 2   | 2   | 2   | 11  | 3   | 4   | 3    | 2   | 2   |
| Attributes (k) | 4   | 5   | 6   | 8   | 9   | 9   | 10  | 13  | 18  | 21   | 34  | 60  |

Table 1: Summary of the datasets used.

the ratio of the test error rate to the baseline rate, which is the error rate of a tree with only a single terminal node.

## 4  The Datasets

Eleven datasets were selected from the database of machine learning tasks maintained by the University of California at Irvine (see the appendix for a list of the datasets used). Except as noted in the appendix, the datasets were used exactly in the form of the distribution as of June 1993. All datasets have only continuous attributes and there are no missing values.[5] The synthetic "waves" example [1] was added as a twelfth dataset.

Table 1 gives a brief description of the datasets. Datasets are identified by a two letter abbreviation along the top. The rows in the table give the total number of instances, number of classes and number of attributes for each dataset.

A few datasets in the original distribution have designated training and testing subsets while others do not. To ensure regularity among datasets, we pooled all usable examples in a given dataset, randomized the order in the pool and then divided the pool into training and testing sets. Two divisions were considered. The *large* training division had $\frac{2}{3}$ of the pooled examples allocated to the training set and $\frac{1}{3}$ to the test set. The *small* training division had $\frac{1}{3}$ of the data in the training set and $\frac{2}{3}$ in the test set.

## 5  Results

Table 2 gives the error rates for both the *large* and *small* divisions of the data, expressed as a percentage of the error rate obtained by guessing the dominant class.

In both the *small* and *large* training divisions of the datasets, the pole-pair method had lower error rates than axis-aligned or linear cart in the majority of datasets tested. While these results are interesting, they do not provide any measure of confidence that one method performs better or worse than another. Since all methods were trained and tested on the same data, we can perform a two-tailed *McNemar test* [2] on the predictions for pairs of methods. The resulting P-values are given in table 3. On most of the tasks, the pole-pair method is significantly better than at least one of the standard methods for at least one of the training set sizes and there are only 2 tasks for which either of the other methods is significantly better on either training set size.

---

determine the best value of $\alpha$ rather than the tree size

[5]In the BC dataset we removed the case identification number attribute and had to delete 16 cases with missing values.

| Database | Small Train | | | Large Train | | |
|----------|------|--------|------|------|--------|------|
|          | cart | linear | pole | cart | linear | pole |
| IR | 14.3 | 14.3 | 4.3 | 5.6 | 5.6 | 5.6 |
| TR | 36.6 | 26.8 | 14.6 | 33.3 | 33.3 | 20.8 |
| LV | 88.9 | 100.0 | 100.0 | 108.7 | 87.0 | 97.8 |
| DB | 85.8 | 82.2 | 87.0 | 69.7 | 69.7 | 59.6 |
| BC | 12.8 | 14.1 | 8.3 | 15.7 | 12.0 | 9.6 |
| GL | 62.5 | 81.3 | 89.6 | 46.4 | 46.4 | 35.7 |
| VW | 31.8 | 37.7 | 30.0 | 21.4 | 26.2 | 19.2 |
| WN | 17.8 | 13.7 | 11.0 | 14.7 | 11.8 | 14.7 |
| VH | 42.5 | 46.5 | 44.2 | 36.2 | 43.9 | 40.7 |
| WV | 28.9 | 25.8 | 24.3 | 30.6 | 24.8 | 26.6 |
| IS | 44.0 | 31.0 | 41.7 | 21.4 | 23.8 | 42.9 |
| SN | 65.2 | 71.2 | 48.5 | 48.4 | 45.2 | 48.4 |

Table 2: Relative error rates expressed as a percentage of the baseline rate on the small and large training sets.

## 6    Discussion

We only considered hyperplanes whose poles were in different classes, since these seemed more plausible candidates. An alternative strategy is to disregard class membership, and consider *all* possible pole-pairs. Another variant of the method arises depending on whether the inputs are scaled. We transformed all inputs so that the training data has zero mean and unit variance. However, using unscaled inputs and/or allowing both poles to have the same class makes little difference to the overall advantage of the pole-pair method.

To summarize, we have demonstrated that the pole-pair method is a simple, effective method for generating projection directions at binary tree nodes. The same idea of minimizing complexity by selecting among a sensible fixed set of possibilities rather than searching a continuous space can also be applied to the choice of input-to-hidden weights in a neural network.

## A    Databases used in the study

IR - Iris plant database.
TR - Thyroid gland data.
LV - BUPA liver disorders.
DB - Pima Indians Diabetes.
BC - Breast cancer database from the University of Wisconsin Hospitals.
GL - Glass identification database. In these experiments we only considered the classification into float/nonfloat processed glass, ignoring other types of glass.
VW - Vowel recognition.
WN - Wine recognition.
VH - Vehicle silhouettes.
WV - Waveform example, the synthetic example from [1].
IS - Johns Hopkins University Ionosphere database.
SN - Sonar - mines versus rocks discrimination. We did not control for aspect-angle.

Small Training - Large Test

| | IR | TR | LV | DB | BC | GL | VW | WN | VH | WV | IS | SN |
|---|---|---|---|---|---|---|---|---|---|---|---|---|
| Axis- Pole | .02 | .02 | .18 | .46 | .06 | .02 | .24 | .15 | .33 | .00 | .44 | .07 |
| Linear- Pole | .02 | .13 | 1.0 | .26 | .02 | .30 | .00 | .41 | .27 | .17 | .09 | .02 |
| Axis-Linear | 1.0 | .06 | .18 | .30 | .40 | .00 | .00 | .31 | .08 | .03 | .02 | .32 |

Large Training - Small Test

| | IR | TR | LV | DB | BC | GL | VW | WN | VH | WV | IS | SN |
|---|---|---|---|---|---|---|---|---|---|---|---|---|
| Axis-Pole | .75 | .23 | .29 | .04 | .11 | .29 | .26 | .69 | .14 | .08 | .02 | .60 |
| Linear-Pole | .75 | .23 | .26 | .04 | .25 | .30 | .01 | .50 | .25 | .26 | .05 | .50 |
| Axis-Linear | 1.0 | 1.0 | .07 | 1.0 | .29 | .69 | .06 | .50 | .03 | .01 | .50 | .50 |

Table 3: P-Values using a two-tailed *McNemar* test on the *small* (top) and *large* (bottom) training sets. Each row gives P-values when the methods in the left most column are compared. A significant difference at the $P = 0.05$ level is indicated with a line above (below) the P-value depending on whether the first (second) mentioned method in the first column had superior performance. For example, in the top most row, the pole-pair method was significantly better than the axis-aligned method on the TR dataset.

## Acknowledgments

We thank Leo Breiman for kindly making his CART code available to us. This research was funded by the Institute for Robotics and Intelligent Systems and by NSERC. Hinton is a fellow of the Canadian Institute for Advanced Research.

## Footnotes

[1] If the threshold value is not midway between the poles, we can still save a factor of $k$ but we need to compute $(d^2_{AC} - d^2_{BC})/2d_{AB}$ instead of just the sign of this expression.

[2] Since we only consider splits in which the poles are in different classes, this number ignores a factor that is independent of $N$.

[3]Impurity is not an information measure but, like an information measure, it is minimized when all the nodes are pure and maximized when all classes at each node have equal probability.

[4]This differs from the conventional application of cross validation, where it is used to

## References

[1] L. Breiman, J. H. Freidman, R. A. Olshen, and C. J. Stone. *Classification and regression trees*. Wadsworth international Group, Belmont, California, 1984.

[2] J. L. Fleiss. *Statistical methods for rates and proportions*. Second edition. Wiley, 1981.

